# Scaling Properties of Coarse-Coded Symbol Memories

Ronald Rosenfeld
David S. Touretzky

Computer Science Department
Carnegie Mellon University
Pittsburgh, Pennsylvania 15213

**Abstract:** Coarse-coded symbol memories have appeared in several neural network symbol processing models. In order to determine how these models would scale, one must first have some understanding of the mathematics of coarse-coded representations. We define the general structure of coarse-coded symbol memories and derive mathematical relationships among their essential parameters: *memory size, symbol-set size* and *capacity*. The computed capacity of one of the schemes agrees well with actual measurements of the coarse-coded working memory of DCPS, Touretzky and Hinton's distributed connectionist production system.

## 1 Introduction

A *distributed representation* is a memory scheme in which each entity (concept, symbol) is represented by a pattern of activity over many units [3]. If each unit participates in the representation of many entities, it is said to be *coarsely tuned*, and the memory itself is called *a coarse-coded memory*.

Coarse-coded memories have been used for storing symbols in several neural network symbol processing models, such as Touretzky and Hinton's distributed connectionist production system DCPS [8,9], Touretzky's distributed implementation of linked list structures on a Boltzmann machine, BoltzCONS [10], and St. John and McClelland's PDP model of case role defaults [6]. In all of these models, memory capacity was measured empirically and parameters were adjusted by trial and error to obtain the desired behavior. We are now able to give a mathematical foundation to these experiments by analyzing the relationships among the fundamental memory parameters.

There are several paradigms for coarse-coded memories. In a *feature-based representation*, each unit stands for some semantic feature. Binary units can code features with binary values, whereas more complicated units or groups of units are required to code more complicated features, such as multi-valued properties or numerical values from a continuous scale. The units that form the representation of a concept define an intersection of features that constitutes that concept. Similarity between concepts composed of binary features can be measured by the Hamming distance between their representations. In a neural network implementation, relationships between concepts are implemented via connections among the units forming their representations. Certain types of generalization phenomena thereby emerge automatically.

A different paradigm is used when representing points in a multidimensional continuous space [2,3]. Each unit encodes values in some subset of the space. Typically the

subsets are hypercubes or hyperspheres, but they may be more coarsely tuned along some dimensions than others [1]. The point to be represented is in the subspace formed by the intersection of all active units. As more units are turned on, the accuracy of the representation improves. The density and degree of overlap of the units' receptive fields determines the system's resolution [7].

Yet another paradigm for coarse-coded memories, and the one we will deal with exclusively, does not involve features. Each concept, or symbol, is represented by an arbitrary subset of the units, called its *pattern*. Unlike in feature-based representations, the units in the pattern bear no relationship to the meaning of the symbol represented. A symbol is stored in memory by turning on all the units in its pattern. A symbol is deemed present if all the units in its pattern are active.[1] The *receptive field* of each unit is defined as the set of all symbols in whose pattern it participates. We call such memories *coarse-coded symbol memories* (CCSMs). We use the term "symbol" instead of "concept" to emphasize that the internal structure of the entity to be represented is not involved in its representation. In CCSMs, a short Hamming distance between two symbols does not imply semantic similarity, and is in general an undesirable phenomenon.

The efficiency with which CCSMs handle sparse memories is the major reason they have been used in many connectionist systems, and hence the major reason for studying them here. The unit-sharing strategy that gives rise to efficient encoding in CCSMs is also the source of their major weakness. Symbols share units with other symbols. As more symbols are stored, more and more of the units are turned on. At some point, some symbol may be deemed present in memory because all of its units are turned on, even though it was not explicitly stored: a "ghost" is born. Ghosts are an unwanted phenomenon arising out of the overlap among the representations of the various symbols. The emergence of ghosts marks the limits of the system's *capacity:* the number of symbols it can store simultaneously and reliably.

## 2 Definitions and Fundamental Parameters

A coarse coded symbol memory in its most general form consists of:

- A set of **N** binary state **units**.

- An alphabet of $\alpha$ **symbols** to be represented. Symbols in this context are atomic entities: they have no constituent structure.

- A **memory scheme**, which is a function that maps each symbol to a subset of the units – its **pattern**. The **receptive field** of a unit is defined as the set of all symbols to whose pattern it belongs (see Figure 1). The exact nature of the

| | $S_1$ | $S_2$ | $S_3$ | $S_4$ | $S_5$ | $S_6$ | $S_7$ | $S8$ |
|---|---|---|---|---|---|---|---|---|
| $U_1$ | • | | | • | • | | • | |
| $U_2$ | | • | • | | • | • | | |
| $U_3$ | | • | | • | • | | | • |
| $U_4$ | • | | | | | • | • | |
| $U_5$ | | | • | | | | | • |
| $U_6$ | • | • | | • | • | | • | |

Figure 1: A memory scheme ($N = 6$, $\alpha = 8$) defined in terms of units $U_i$ and symbols $S_j$. The columns are the symbols' *patterns*. The rows are the units' *receptive fields*.

memory scheme mapping determines the properties of the memory, and is the central target of our investigation.

As symbols are stored, the memory fills up and ghosts eventually appear. It is not possible to detect a ghost simply by inspecting the contents of memory, since there is no general way of distinguishing a symbol that was stored from one that emerged out of overlaps with other symbols. (It is sometimes possible, however, to conclude that there are no ghosts.) Furthermore, a symbol that emerged as a ghost at one time may not be a ghost at a later time if it was subsequently stored into memory. Thus the definition of a ghost depends not only on the state of the memory but also on its history.

Some memory schemes guarantee that no ghost will emerge as long as the number of symbols stored does not exceed some specified limit. In other schemes, the emergence of ghosts is an ever-present possibility, but its probability can be kept arbitrarily low by adjusting other parameters. We analyze systems of both types. First, two more bits of notation need to be introduced:

$P_{\text{ghost}}$: **Probability of a ghost.** The probability that at least one ghost will appear after some number of symbols have been stored.

k: **Capacity.** The maximum number of symbols that can be stored simultaneously before the probability of a ghost exceeds a specified threshold. If the threshold is 0, we say that the capacity is **guaranteed.**

A localist representation, where every symbol is represented by a single unit and every unit is dedicated to the representation of a single symbol, can now be viewed as a special case of coarse-coded memory, where $k = N = \alpha$ and $P_{\text{ghost}} = 0$. Localist representations are well suited for memories that are not sparse. In these cases, coarse-coded memories are at a disadvantage. In designing coarse-coded symbol memories we are interested in cases where $k \ll N \ll \alpha$. The permissible probability for a ghost in these systems should be low enough so that its impact can be ignored.

## 3   Analysis of Four Memory Schemes

### 3.1   Bounded Overlap (guaranteed capacity)

If we want to construct the memory scheme with the largest possible $\alpha$ (given $N$ and $k$) while guaranteeing $P_{\text{ghost}} = 0$, the problem can be stated formally as:

> Given a set of size $N$, find the largest collection of subsets of it such that no union of $k$ such subsets subsumes any other subset in the collection.

This is a well known problem in Coding Theory, in slight disguise. Unfortunately, no complete analytical solution is known. We therefore simplify our task and consider only systems in which all symbols are represented by the same number of units (i.e. all patterns are of the same size). In mathematical terms, we restrict ourselves to constant weight codes. The problem then becomes:

> Given a set of size $N$, find the largest collection of subsets of size *exactly* $L$ such that no union of $k$ such subsets subsumes any other subset in the collection.

There are no known complete analytical solutions for the size of the largest collection of patterns even when the patterns are of a fixed size. Nor is any efficient procedure for constructing such a collection known. We therefore simplify the problem further. We now restrict our consideration to patterns whose pairwise overlap is bounded by a given number. For a given pattern size $L$ and desired capacity $k$, we require that no two patterns overlap in more than $m$ units, where:

$$m = \left\lfloor \frac{L-1}{k} \right\rfloor \tag{1}$$

Memory schemes that obey this constraint are guaranteed a capacity of at least $k$ symbols, since any $k$ symbols taken together can overlap at most $L - 1$ units in the pattern of any other symbol – one unit short of making it a ghost. Based on this constraint, our mathematical problem now becomes:

> Given a set of size $N$, find the largest collection of subsets of size exactly $L$ such that the intersection of any two such subsets is of size $\leq m$ (where $m$ is given by equation 1.)

Coding theory has yet to produce a complete solution to this problem, but several methods of deriving upper bounds have been proposed (see for example [4]). The simple formula we use here is a variant of the Johnson Bound. Let $\alpha_{bo}$ denote the maximum number of symbols attainable in memory schemes that use bounded overlap. Then

$$\alpha_{bo}(N, L, m) \quad \leq \quad \frac{\binom{N}{m+1}}{\binom{L}{m+1}} \tag{2}$$

The Johnson bound is known to be an *exact* solution asymptotically (that is, when $N, L, m \rightarrow \infty$ and their ratios remain finite).

Since we are free to choose the pattern size, we optimize our memory scheme by maximizing the above expression over all possible values of $L$. For the parameter subspace we are interested in here ($N < 1000$, $k < 50$) we use numerical approximation to obtain:

$$\alpha_{bo}(N,k) = \max_{L \in [1,N]} \frac{\binom{N}{m+1}}{\binom{L}{m+1}} < \max_{L \in [1,N]} \left(\frac{N}{L-m}\right)^{m+1} < e^{0.367\frac{N}{k}} \quad (3)$$

(Recall that $m$ is a function of $L$ and $k$.) Thus the upper bound we derived depicts a simple exponential relationship between $\alpha$ and $N/k$. Next, we try to construct memory schemes of this type. A Common Lisp program using a modified depth-first search constructed memory schemes for various parameter values, whose $\alpha$'s came within 80% to 90% of the upper bound. These results are far from conclusive, however, since only a small portion of the parameter space was tested.

In evaluating the viability of this approach, its apparent optimality should be contrasted with two major weaknesses. First, this type of memory scheme is hard to construct computationally. It took our program several minutes of CPU time on a Symbolics 3600 to produce reasonable solutions for cases like $N = 200, k = 5, m = 1$, with an exponential increase in computing time for larger values of $m$. Second, if CC-SMs are used as models of memory in naturally evolving systems (such as the brain), this approach places too great a burden on developmental mechanisms.

The importance of the bounded overlap approach lies mainly in its role as an upper bound for all possible memory schemes, subject to the simplifications made earlier. All schemes with guaranteed capacities can be measured relative to equation 3.

## 3.2 Random Fixed Size Patterns (a stochastic approach)

Randomly produced memory schemes are easy to implement and are attractive because of their naturalness. However, if the patterns of two symbols coincide, the guaranteed capacity will be zero (storing one of these symbols will render the other a ghost). We therefore abandon the goal of guaranteeing a certain capacity, and instead establish a tolerance level for ghosts, $P_{\text{ghost}}$. For large enough memories, where stochastic behavior is more robust, we may expect reasonable capacity even with very small $P_{\text{ghost}}$.

In the first stochastic approach we analyze, patterns are randomly selected subsets of a fixed size $L$. Unlike in the previous approach, choosing $k$ does not bound $\alpha$. We may define as many symbols as we wish, although at the cost of increased probability of a ghost (or, alternatively, decreased capacity). The probability of a ghost appearing after $k$ symbols have been stored is given by Equation 4:

$$P_{\text{ghost}}(N,L,k,\alpha) = 1 - \sum_{c=L}^{\min(N,kL)} T_{N,L}(k,c) \cdot \left[1 - \frac{\binom{C}{L}}{\binom{N}{L}}\right]^{\alpha-k} \quad (4)$$

$T_{N,L}(k,c)$ is the probability that exactly $c$ units will be active after $k$ symbols have been stored. It is defined recursively by Equation 5:

$$T_{N,L}(0,0) = 1$$
$$T_{N,L}(k,c) = 0 \quad \text{for either } k = 0 \text{ and } c \neq 0, \text{ or } k > 0 \text{ and } c < L \qquad (5)$$
$$T_{N,L}(k,c) = \sum_{a=0}^{L} T(k-1, c-a) \cdot \binom{N-(c-a)}{a} \cdot \binom{c-a}{L-a} / \binom{N}{L}$$

We have constructed various coarse-coded memories with random fixed-size receptive fields and measured their capacities. The experimental results show good agreement with the above equation.

The optimal pattern size for fixed values of $N$, $k$, and $\alpha$ can be determined by binary search on Equation 4, since $P_{\text{ghost}}(L)$ has exactly one maximum in the interval $[1, N]$. However, this may be expensive for large $N$. A computational shortcut can be achieved by estimating the optimal $L$ and searching in a small interval around it. A good initial estimate is derived by replacing the summation in Equation 4 with a single term involving $E[c]$: the expected value of the number of active units after $k$ symbols have been stored. The latter can be expressed as:

$$E[c] = N \cdot \left[ 1 - (1 - L/N)^k \right]$$

The estimated $L$ is the one that maximizes the following expression:

$$\binom{E[c]}{L} / \binom{N}{L}$$

An alternative formula, developed by Joseph Tebelskis, produces very good approximations to Eq. 4 and is much more efficient to compute. After storing $k$ symbols in memory, the probability $P_z$ that a single arbitrary symbol $x$ has become a ghost is given by:

$$P_z(N, L, k, \alpha) = \sum_{j=0}^{L} (-1)^j \binom{L}{j} \binom{N-j}{L}^k / \binom{N}{L}^k \qquad (6)$$

If we now assume that each symbol's $P_z$ is independent of that of any other symbol, we obtain:

$$P_{\text{ghost}} \approx 1 - (1 - P_z)^{\alpha - k} \qquad (7)$$

This assumption of independence is not strictly true, but the relative error was less than 0.1% for the parameter ranges we considered, when $P_{\text{ghost}}$ was no greater than 0.01.

We have constructed the two-dimensional table $T_{N,L}(k,c)$ for a wide range of $(N, L)$ values ($70 \leq N \leq 1000$, $7 \leq L \leq 43$), and produced graphs of the relationships between $N$, $k$, $\alpha$, and $P_{\text{ghost}}$ for optimum pattern sizes, as determined by Equation 4. The

results show an approximately exponential relationship between $\alpha$ and $N/k$ [5]. Thus, for a fixed number of symbols, the capacity is proportional to the number of units. Let $\alpha_{rfp}$ denote the maximum number of symbols attainable in memory schemes that use random fixed-size patterns. Some typical relationships, derived from the data, are:

$$
\begin{aligned}
\alpha_{rfp}(P_{\text{ghost}} = 0.01) &\approx 0.0086 \cdot e^{0.468 \frac{N}{k}} \\
\alpha_{rfp}(P_{\text{ghost}} = 0.001) &\approx 0.0008 \cdot e^{0.473 \frac{N}{k}}
\end{aligned}
\tag{8}
$$

### 3.3 Random Receptors (a stochastic approach)

A second stochastic approach is to have each unit assigned to each symbol with an independent fixed probability $s$. This method lends itself to easy mathematical analysis, resulting in a closed-form analytical solution.

After storing $k$ symbols, the probability that a given unit is active is $1 - (1 - s)^k$ (independent of any other unit). For a given symbol to be a ghost, every unit must either be active or else not belong to that symbol's pattern. That will happen with a probability $\left[1 - s \cdot (1 - s)^k\right]^N$, and thus the probability of a ghost is:

$$
P_{\text{ghost}}(\alpha, N, k, s) = 1 - \left[1 - \left[1 - s \cdot (1 - s)^k\right]^N\right]^{\alpha - k}
\tag{9}
$$

Assuming $P_{\text{ghost}} \ll 1$ and $k \ll \alpha$ (both hold in our case), the expression can be simplified to:

$$
P_{\text{ghost}}(\alpha, N, k, s) = \alpha \cdot \left[1 - s \cdot (1 - s)^k\right]^N
$$

from which $\alpha$ can be extracted:

$$
\alpha_{rr}(N, k, s, P_{\text{ghost}}) = \frac{P_{\text{ghost}}}{\left[1 - s \cdot (1 - s)^k\right]^N}
\tag{10}
$$

We can now optimize by finding the value of $s$ that maximizes $\alpha$, given any desired upper bound on the expected value of $P_{\text{ghost}}$. This is done straightforwardly by solving $\partial \alpha / \partial s = 0$. Note that $s \cdot N$ corresponds to $L$ in the previous approach. The solution is $s = 1/(k+1)$, which yields, after some algebraic manipulation:

$$
\alpha_{rr} = P_{\text{ghost}} \cdot e^{N \log\left[(k+1)^{k+1} / \left((k+1)^{k+1} - k^k\right)\right]}
\tag{11}
$$

A comparison of the results using the two stochastic approaches reveals an interesting similarity. For large $k$, with $P_{\text{ghost}} = 0.01$ the term $0.468/k$ of Equation 8 can be seen as a numerical approximation to the log term in Equation 11, and the multiplicative factor of 0.0086 in Equation 8 approximates $P_{\text{ghost}}$ in Equation 11. This is hardly surprising, since the Law of Large Numbers implies that in the limit ($N, k \to \infty$, with $s$ fixed) the two methods are equivalent.

Finally, it should be noted that the stochastic approaches we analyzed generate a family of memory schemes, with non-identical ghost-probabilities. $P_{\text{ghost}}$ in our formulas is therefore better understood as *an expected value,* averaged over the entire family.

## 3.4 Partitioned Binary Coding (a reference point)

The last memory scheme we analyze is not strictly distributed. Rather, it is somewhere in between a distributed and a localist representation, and is presented for comparison with the previous results. For a given number of units $N$ and desired capacity $k$, the units are partitioned into $k$ equal-size "slots," each consisting of $N/k$ units (for simplicity we assume that $k$ divides $N$). Each slot is capable of storing exactly one symbol.

The most efficient representation for all possible symbols that may be stored into a slot is to assign them binary codes, using the $N/k$ units of each slot as bits. This would allow $2^{N/k}$ symbols to be represented. Using binary coding, however, will not give us the required capacity of 1 symbol, since binary patterns subsume one another. For example, storing the code '10110' into one of the slots will cause the codes '10010', '10100' and '00010' (as well as several other codes) to become ghosts.

A possible solution is to use only half of the bits in each slot for a binary code, and set the other half to the binary complement of that code (we assume that $N/k$ is even). This way, the codes are guaranteed not to subsume one another. Let $\alpha_{pbc}$ denote the number of symbols representable using a partitioned binary coding scheme. Then,

$$\alpha_{pbc} = 2^{N/2k} = e^{0.347\frac{N}{k}} \tag{12}$$

Once again, $\alpha$ is exponential in $N/k$. The form of the result closely resembles the estimated upper bound on the Bounded Overlap method given in Equation 3. There is also a strong resemblance to Equations 8 and 11, except that the fractional multiplier in front of the exponential, corresponding to $P_{\text{ghost}}$, is missing. $P_{\text{ghost}}$ is 0 for the Partitioned Binary Coding method, but this is enforced by dividing the memory into disjoint sets of units rather than adjusting the patterns to reduce overlap among symbols.

As mentioned previously, this memory scheme is not really distributed in the sense used in this paper, since there is no one pattern associated with a symbol. Instead, a symbol is represented by any one of a set of $k$ patterns, each $N/k$ bits long, corresponding to its appearance in one of the $k$ slots. To check whether a symbol is present, all $k$ slots must be examined. To store a new symbol in memory, one must scan the $k$ slots until an empty one is found. Equation 12 should therefore be used only as a point of reference.

## 4 Measurement of DCPS

The three distributed schemes we have studied all use unstructured patterns, the only constraint being that patterns are at least roughly the same size. Imposing more complex structure on any of these schemes may is likely to reduce the capacity somewhat. In

| Memory Scheme | Result |
|---|---|
| Bounded Overlap | $\alpha_{bo}(N, k) \quad < \quad e^{0.367\frac{N}{k}}$ |
| Random Fixed-size Patterns | $\alpha_{rfp}(P_{\text{ghost}} = \quad 0.01) \approx 0.0086 \cdot e^{0.468\frac{N}{k}}$ |
|  | $\alpha_{rfp}(P_{\text{ghost}} = 0.001) \approx 0.0008 \cdot e^{0.473\frac{N}{k}}$ |
| Random Receptors | $\alpha_{rr} = P_{\text{ghost}} \cdot e^{N \cdot \log (k+1)^{k+1}/((k+1)^{k+1} - k^k)}$ |
| Partitioned Binary Coding | $\alpha_{pbc} \quad = \quad e^{0.347\frac{N}{k}}$ |

Table 1 Summary of results for various memory schemes.

order to quantify this effect, we measured the memory capacity of DCPS (BoltzCONS uses the same memory scheme) and compared the results with the theoretical models analyzed above.

DCPS' memory scheme is a modified version of the Random Receptors method [5]. The symbol space is the set of all triples over a 25 letter alphabet. Units have fixed-size receptive fields organized as $6 \times 6 \times 6$ subspaces. Patterns are manipulated to minimize the variance in pattern size across symbols. The parameters for DCPS are: $N = 2000$, $\alpha = 25^3 = 15625$, and the mean pattern size is $(6/25)^3 \times 2000 = 27.65$ with a standard deviation of 1.5. When $P_{\text{ghost}} = 0.01$ the measured capacity was $k = 48$ symbols. By substituting for $N$ in Equation 11 we find that the highest $k$ value for which $\alpha_{rr} \geq 15625$ is 51. There does not appear to be a significant cost for maintaining structure in the receptive fields.

## 5 Summary and Discussion

Table 1 summarizes the results obtained for the four methods analyzed. Some differences must be emphasized:

- $\alpha_{bo}$ and $\alpha_{pbc}$ deal with guaranteed capacity, whereas $\alpha_{rfp}$ and $\alpha_{rr}$ are meaningful only for $P_{\text{ghost}} > 0$.

- $\alpha_{bo}$ is only an upper bound.

- $\alpha_{rfp}$ is based on numerical estimates.

- $\alpha_{pbc}$ is based on a scheme which is not strictly coarse-coded.

The similar functional form of all the results, although not surprising, is aesthetically pleasing. Some of the functional dependencies among the various parameters can be derived informally using qualitative arguments. Only a rigorous analysis, however, can provide the definite answers that are needed for a better understanding of these systems and their scaling properties.

## Acknowledgments

We thank Geoffrey Hinton, Noga Alon and Victor Wei for helpful comments, and Joseph Tebelskis for sharing with us his formula for approximating $P_{ghost}$ in the case of fixed pattern sizes.

This work was supported by National Science Foundation grants IST-8516330 and EET-8716324, and by the Office of Naval Research under contract number N00014-86-K-0678. The first author was supported by a National Science Foundation graduate fellowship.

## Footnotes

[1]This criterion can be generalized by introducing a *visibility threshold*: a fraction of the pattern that should be on in order for a symbol to be considered present. Our analysis deals only with a visibility criterion of 100%, but can be generalized to accommodate noise.

## References

[1] Ballard, D H. (1986) Cortical connections and parallel processing: structure and function. *Behavioral and Brain Sciences* 9(1).

[2] Feldman, J. A., and Ballard, D. H. (1982) Connectionist models and their properties. *Cognitive Science* 6, pp. 205-254.

[3] Hinton, G. E., McClelland, J. L., and Rumelhart, D. E. (1986) Distributed representations. In D. E. Rumelhart and J. L. McClelland (eds.), *Parallel Distributed Processing: Explorations in the Microstructure of Cognition*, volume 1. Cambridge, MA: MIT Press.

[4] Macwilliams, F.J., and Sloane, N.J.A. (1978). *The Theory of Error-Correcting Codes*, North-Holland.

[5] Rosenfeld, R. and Touretzky, D. S. (1987) Four capacity models for coarse-coded symbol memories. Technical report CMU-CS-87-182, Carnegie Mellon University Computer Science Department, Pittsburgh, PA.

[6] St. John, M. F. and McClelland, J. L. (1986) Reconstructive memory for sentences: a PDP approach. *Proceedings of the Ohio University Inference Conference*.

[7] Sullins, J. (1985) Value cell encoding strategies. Technical report TR-165, Computer Science Department, University of Rochester, Rochester, NY.

[8] Touretzky, D. S., and Hinton, G. E. (1985) Symbols among the neurons: details of a connectionist inference architecture. *Proceedings of IJCAI-85*, Los Angeles, CA, pp. 238-243.

[9] Touretzky, D. S., and Hinton, G. E. (1986) A distributed connectionist production system. Technical report CMU-CS-86-172, Computer Science Department, Carnegie Mellon University, Pittsburgh, PA.

[10] Touretzky, D. S. (1986) BoltzCONS: reconciling connectionism with the recursive nature of stacks and trees. *Proceedings of the Eighth Annual Conference of the Cognitive Science Society*, Amherst, MA, pp. 522-530.